# Active Gesture Recognition using Learned Visual Attention

**Trevor Darrell and Alex Pentland**
Perceptual Computing Group
MIT Media Lab
20 Ames Street, Cambridge MA, 02138
trevor,sandy@media.mit.edu

## Abstract

We have developed a foveated gesture recognition system that runs in an unconstrained office environment with an active camera. Using vision routines previously implemented for an interactive environment, we determine the spatial location of salient body parts of a user and guide an active camera to obtain images of gestures or expressions. A hidden-state reinforcement learning paradigm is used to implement visual attention. The attention module selects targets to foveate based on the goal of successful recognition, and uses a new multiple-model Q-learning formulation. Given a set of target and distractor gestures, our system can learn where to foveate to maximally discriminate a particular gesture.

## 1 INTRODUCTION

Vision has numerous uses in the natural world. It is used by many organisms in navigation and object recognition tasks, for finding resources or avoiding predators. Often overlooked in computational models of vision, however, and particularly relevant for humans, is the use of vision for communication and interaction. In these domains visual perception is an important communication modality, either in addition to language or when language cannot be used. In general, people place considerable weight on visual signals from another individual, such as facial expression, hand gestures, and body language. We have been developing neurally-inspired methods which combine low-level vision and learning to model these visual abilities.

Previously, we presented a method for view-based recognition of spatio-temporal hand gestures [2] and a similar mechanism for the analysis/real-time tracking of facial expressions [4]. These methods offered real-time performance and a relatively high level of accuracy, but required foveated images of the object performing the

gesture. There are many domains/tasks for which these are not unreasonable assumptions, such as interaction with a single user workstation or an automobile with a single driver. However the method had limited usefulness in unconstrained domains, such as "intelligent rooms" or interactive virtual environments, when the identity and location of the user are unknown.

In this paper, we expand our gesture recognition method to include an active component, utilizing a foveated image sensor that can selectively track a person's hand or face as they walk through a room. The camera tracking and model selection routines are guided by an action-selection system that implements visual attention based on reinforcement learning. Using on a simple reward schedule, this attention system learns the appropriate object (hand, head) to foveate in order to maximize recognition performance.

## 2  FOVEATED GESTURE ANALYSIS

Our system for foveated gesture recognition combines person tracking routines, an active, high-resolution camera, and view-based normalized correlation analysis. First we will briefly describe the person tracking module and view-based analysis, then discuss their use with an active camera.

We have implemented vision routines to track a user in in an office setting as part of our ALIVE system, an Artificial Life Interactive Video Environment[3]. This system can track people and identify head/hand locations as they walk about a room, and provides the contextual environment within which view-based gesture analysis methods can be successfully applied. The ALIVE system assumed little prior knowledge of the user, and operated on coarse-scale images.[1] ALIVE allows a user to interact with virtual artificial life creatures, through the use of a "magic-mirror" metaphor in which user sees him/herself presented in a video display along with virtual creatures. A wide field-of-view video camera acquires an image of the user, which is then combined with computer graphics imagery and projected on a large screen in front of the user. Vision routines in ALIVE compute figure/ground segmentation and analyze the user's silhouette to determine the location of head, hands, and other salient body features. We use only a single, calibrated, wide field-of-view camera to determine the 3-D position of these features.[2] For details of our person tracking method see [14].

In our approach to real-time expression matching/tracking, a set of view-based correlation models is used to represent spatio-temporal gesture patterns. We take a sequence of images representing the gesture to be trained, and build a set of view models that are sufficient to track the object as it performs the gesture. Our view models are normalized correlation templates, and can either be intensity-based or based on band-pass or wavelet-based signal representations.[3] We applied our model to the problem of hand gesture recognition [2] as well as for tracking facial expressions [4]. For facial tracking, we implemented an interpolation paradigm to map view-based correlation scores to facial motor controls. We used the Radial Basis Function (RBF) method[7]; interpolation was performed using a set of exemplars consisting of pairs of real faces and model faces in different expressions, which were

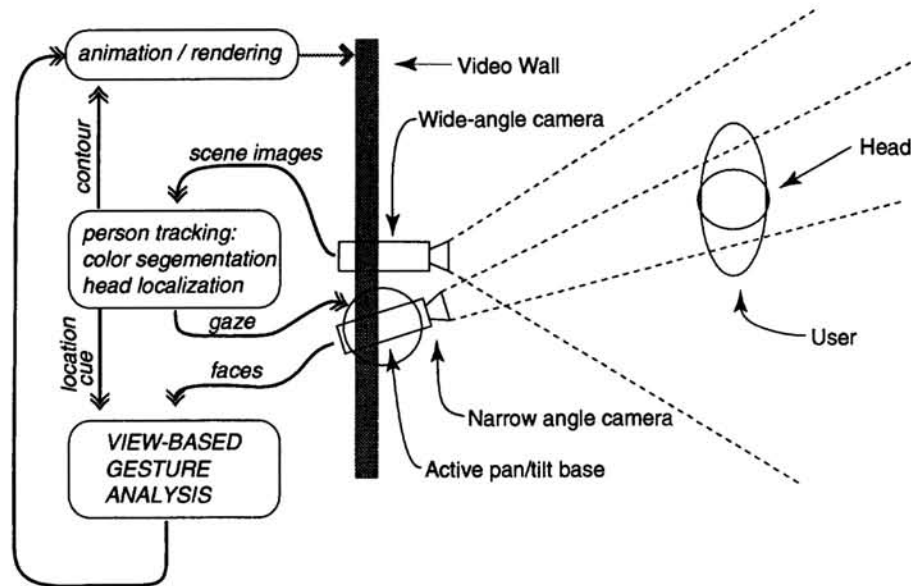

Figure 1: Overview of system for person tracking and active gesture recognition. Static, wide-field-of-view, camera tracks user's head and hands, which drives gaze control of active narrow-field-of-view camera. Foveated images are used for view-based gesture analysis and recognition. Graphical objects are rendered on video wall and can react to user's position, pose, and gestures.

obtained by generating a 3-D model face and asking the user to match it. With this simple formalism, we were able to track expressions of a real user and interpolate equivalent 3-D model faces in real-time.

This view-based analysis requires detailed imagery, which cannot be obtained from a single, fixed camera as the user walks about a room. To provide high resolution images for gesture recognition, we augment the wide field-of-view camera in our interactive environment with an active, narrow-field-of-view camera, as shown in Figure 1. Information about head/hand location from the existing ALIVE routines is used to drive the motor control parameters of the narrow field camera. Currently the camera can be directed to autonomously track head or hands. Using a highly simplified, two expression model of facial expression (neutral and surprised), we have been able to track facial expressions as users move about the room and the narrow angle camera followed the face. For details on this foveated gesture recognition see [5]

## 3   VISUAL ATTENTION FOR RECOGNITION

The visual routines in the ALIVE system can be used to track the head and hands of a user, and the active camera can provide foveated images for gesture recognition. If we know a priori which body part will produce the gesture of interest, or if we have a sufficient number of active cameras to track all body parts, then we have solved the problem. Of course, in practice there are more possible loci of gesture performance than there are active cameras, and we have to address the problem of action selection for visual routines, i.e., attention. In our active gesture recognition system, we have adopted an action selection model based on reinforcement learning.

## 3.1   THE ACTIVE GESTURE RECOGNITION PROBLEM

We define an Active Gesture Recognition (AGR) task as follows. First, we assume primitive routines exist to provide the continuous valued control and tracking of the different body parts that perform gestures. Second, we assume that body pose and hand/face state is represented as a feature set, based on the representation produced by our body tracker and view-based recognition system, and we define a gesture to be a configuration of the user's body pose and hand/face expression. Third, we assume that, in addition to there being actions for foveating all the relevant body parts, there is also a special action labeled accept, and that the execution of this action by the AGR system signifies detection of the gesture. Finally, the goal of the AGR task is to execute the accept action whenever the user is in the target gesture state, and not to perform that action when the user is in any other (e.g. distractor) state. The AGR system should use the foveation actions to optimally discriminate the target pattern from distractor patterns, even when no single view of the user is sufficient to decide what gesture the user is performing.

An important problem in applying reinforcement learning to this task is that our perceptual observations may not provide a complete description of the user's state. Indeed, because we have a foveated image sensor we know that the user's true gestural state will be hidden whenever the user is performing a gesture and the camera is not foveated on the appropriate body part. By definition, a system for perceptual action selection must not assume a full observation of state is available, otherwise there would be no meaningful perception taking place.

The AGR task can be considered as a Partially Observable Markov Decision Process (POMDP), which is essentially a Markov Decision Process without direct access to state[11, 9]. Rather than attempt to solve them explicitly, we look to techniques for hidden state reinforcement learning to find a solution [10, 8, 6, 1]. A POMDP consists of a set of states in the world $S$, a set of observations $O$, a set of actions $A$, a reward function $R$. After executing an action $a$, the likelihood of transitioning between two states $s, s'$ is given by $T(s, a, a')$, an observation $o$ is generated with probability $O(s, a, o)$. In practice, $T$ and $O$ are not easily obtainable, and we use reinforcement learning methods which do not require them *a priori*.

Our state is defined by the users pose, facial expression, and hand configurations, expressed in nine variables. Three are boolean and are provided directly by the person tracker: person-present, left-arm-extended, and right-arm-extended. Three more are provided by the foveated gesture recognition system, (face, left-hand, right-hand), and take on an integer number of values according to the number of view-based expressions/hand-poses: in our first experiments face can be one of neutral, smile, or surprise, and the hands can each be one of neutral, point, or grab. In addition, three boolean features represent the internal state of the vision system: head-foveated, left-hand-foveated, right-hand-foveated. At each time step, the world is defined by a state $s \in S$, which is defined by these features. An observation, $o \in O$, consists of the same feature variables, except that those provided by the foveated gesture system (e.g., head and hands) are only observable when foveated. Thus the face variable is hidden unless the head-foveated variable is set, the left-hand variable hidden unless the left-hand-foveated variable set, and similarly with the right hand. Hidden variables are set to a undefined value.

The set of actions, $A$, available to the AGR system are 4 foveation commands: look-body, look-head, look-left-hand, and look-right-hand plus the special accept action. Each foveation command causes the active camera to follow the respective body part, and sets the internal foveation feature bits accordingly.

The reward function provides a unit positive reward whenever the **accept** action is performed and the user is in the target state (as defined by an oracle, external to the AGR system), and a fixed negative reward of magnitude $\alpha$ when performed and the user is in a distractor (non-target) state. Zero reward is given whenever a foveation action is performed.

## 3.2  HIDDEN-STATE REINFORCEMENT LEARNING

We have implemented a instance-based method for hidden state reinforcement learning, based on earlier work by McCallum [10]. The instance-based approach to reinforcement learning replaces the absolute state with a distributed memory-based state representation. Given a history of action, reward, and observation tuples, $(a[t], r[t], o[t])$, $0 \leq t \leq T$, a Q-value is also stored with each time step, $q[t]$, and Q-learning[12, 13] is performed by evaluating the similarity of recently observed tuples with sequences farther back in the history chain. Q-values are computed, and the Q-learning update rule applied, maintaining this distributed, memory-based representation of Q-values.

As in traditional Q-learning, at each time step the utility of each action in the current state is evaluated. If full access to the state was available and a table used to represent Q values, this would simply be a table look-up operation, but in a POMDP we do not have full access to state. Using a variation on the instance based approach employed by McAllum's Nearest Sequence Memory (NSM) algorithm, we instead find the $K$ nearest neighbors in the history list relative to the current time point, and compute their average Q value. For each element on the history list, we compute the sequence match criteria with the current time point, $M(i, T)$, where

$$M(i, j) = S(i, j) + M(i - 1, j - 1) \quad \text{if } S(i, j) > 0 \text{ and } i > 0 \text{ and } j > 0$$

$$0 \quad \text{otherwise} \ .$$

We define $S(i, j)$ to be 1 if $o[i] = o[j]$ or $a[i] = a[j]$, 2 if both are equal, and 0 otherwise. Using a superscript in parentheses to denote the action index of a Q-value, we then compute

$$Q^{(a)}[T] = (1/K) \sum_{i=0}^{T} v^{(a)}[i]q[t] \ , \tag{1}$$

where $v^{(a^*)}[i]$ indicates whether the history tuple at time step $i$ votes when computing the Q-value of a new action $a^*$: $v^{(a^*)}[i]$ is set to 1 when $a[i] = a^*$ and $M(i-1, T)$ is among the $K$ largest match values for all $k$ which have $a[k] = a^*$, otherwise it is set to 0. Given Q values for each action the optimal policy is simply

$$\pi[T] = \arg\max_{a \in \mathcal{A}} Q^{(a)}[T] \ . \tag{2}$$

The new action $a[T + 1]$ is chosen either according to this policy or based on an exploration strategy. In either case, the action is executed yielding an observation and reward, and a new tuple added to the history. The new Q-value is set to be the Q value of the chosen action, $q[T + 1] = Q^{(a[T+1])}[T]$. The update step of Q learning is then computed, evaluating

$$U[T + 1] = \max_{a \in \mathcal{A}} Q^{(a)}[T + 1] \ , \tag{3}$$

$$q[i] \leftarrow (1 - \beta)q[i] + \beta(r[i] + \gamma U[T + 1]) \ , \tag{4}$$

for each $i$ such that $v^{(a[T+1])}[i] = 1$.

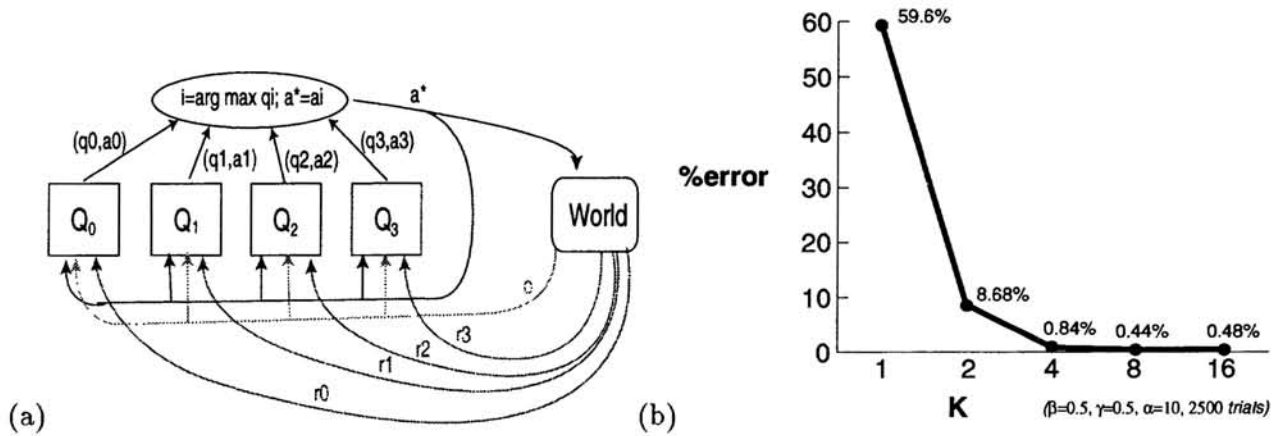

(a)                                                                            (b)

Figure 2: (a) Multiple model Q-learning: one Q-learning agent for each target gesture to be recognized, with coupled observation and action but separate reward and Q-value. (b) Results on recognition task with 8 gesture targets; graph shows error rate after convergence plotted as a function of number of nearest neighbors used in learning algorithm.

## 4 MULTIPLE MODEL Q-LEARNING

In general, we have found the simple, instance-based hidden state reinforcement learning described above to be an effective way to perform action selection for foveation when the task is recognition of a single object from a set of distractors. However, we did not find that this type of system performed well when the AGR task was extended to include more than one target gesture. When multiple **accept** actions were added to enumerate the different targets, we were not able to find exploration strategies that would converge in reasonable time.

This is not unexpected, since the addition of multiple causes of positive reward makes the Q-value space considerably more complex. To remedy this problem, we propose a multiple model Q-learning system. In a multiple model approach to the AGR problem, separate learning agents model the task from each targets perspective. Conceptually, a separate Q-learning agent exists for each target, maintains it's own Q-value and history structure, and is coupled to the other agents via shared observations. Since we can interpret the Q-value of an individual AGR agent as a confidence value that its target is present, we can mediate among the actions predicted by the different agents by selecting the action from the agent with highest Q-value (Figure 2).

Formally, in our multiple model Q-learning system all agents share the same observation and selected action, but have different reward and Q-values. Thus they can be considered a single Q-learning system, but with vector reward and Q-values. Our multiple model learning system is thus obtained by rewriting Eqs. (1)-(4) with vector $q[t]$ and $r[t]$. Using a subscript $j$ to indicate the target index, we have

$$Q_j^{(a)}[T] = (1/K) \sum_{i=0}^{T} v^{(a)}[i] q_j[t] , \quad \pi[T] = \arg \max_{a \in \mathcal{A}} \left( \max_j Q_j^{(a)}[T] \right) . \quad (5)$$

Rewards are computed with: if $a[T] = $ **accept** then $r_j[T] = R(j,T)$ else $r_j[T] = 0$; $R(j,T) = 1$ if gesture $j$ was present at time $T$, else $R(j,T) = -\alpha$. Further,

$$U_j[T+1] = \max_{a \in \mathcal{A}} Q_j^{(a)}[T+1] , \quad (6)$$

$$q_j[i] \leftarrow (1 - \beta)q_j[i] + \beta(r_j[i] + \gamma U_j[T+1]) \quad \forall \ i \ \text{s.t.} \ v^{(a[T+1])}[i] = 1 \ . \tag{7}$$

Note that our sequence match criteria, unlike that in [10], does not depend on $r[t]$; this allows considerable computational savings in the multiple model system since $v^{(a)}$ need not depend on $j$.

We ran the multiple model learning system on the AGR task using 8 targets, with $\beta = 0.5, \gamma = 0.5, \alpha = 10$. Results summed over 2500 trials are shown in Figure 2(b), with classification error plotted against the number of nearest neighbors used in the NSM algorithm. The error rate shown is after convergence; we ran the algorithm with a period of deterministic exploration before following the optimal policy. (The system deterministically explored each action/accept pair.) As can be seen from the graph, for any non-degenerate value of $K$ reasonable performance was obtained; for $K > 2$, the system performed almost perfectly.

## Footnotes

[1]A simple mechanism for recognition of hand gestures was implemented in the original ALIVE system but made no use of high-resolution view models, and could only recognize pointing and waving motions defined by the motion of the centroid of the hand.

[2]By assuming the the user is sitting or standing on the ground plane, we use the imaging and ground plane geometry to compute the location of the user in 3-D.

[3]The latter have the advantage of being less dependent on illumination direction.

# References

[1] A. Cassandra, L. P. Kaelbling, and M. Littman. Acting optimally in partially observable stochastic domains. In *Proc. AAAI-94*, pages 1023–1028. Morgan Kaufmann, 1994.

[2] T. Darrell and A. P. Pentland. Classification of Hand Gestures using a View-Based Distributed Representation In *Advances in Neural Information Processing Systems 6*, Morgan Kauffman, 1994.

[3] T. Darrell, P. Maes, B. Blumberg, and A. P. Pentland, A Novel Environment for Situated Vision and Behavior, *Proc. IEEE Workshop for Visual Behaviors*, IEEE Comp. Soc. Press, Los Alamitos, CA, 1994

[4] T. Darrell, I. Essa, and A. P. Pentland, Correlation and Interpolation Networks for Real-time Expression Analysis/Synthesis, In *Advances in Neural Information Processing Systems 7*, MIT Press, 1995.

[5] T. Darrell and A. Pentland, A., Attention-driven Expression and Gesture Analysis in an Interactive Environment, in *Proc. Intl. Workshop on Automatic Face and Gesture Recognition (IWAFGR '95)*, Zurich, Switzerland, 1995.

[6] T. Jaakkola, S. Singh, and M. Jordan. Reinforcement Learning Algorithm for Partially Observable Markov Decision Problems. In *Advances In Neural Information Processing Systems 7*, MIT Press, 1995.

[7] T. Poggio and F. Girosi, A Theory of Networks for Approximation and Learning. MIT AI Lab TR-1140, 1989.

[8] L. Lin and T. Michell. Reinforcement learning with hidden states. In *Proc. AAAI-92*. Morgan Kaufmann, 1992.

[9] W. Lovejoy. A survey of algorithmic methods of partially observed markov decision processes. *Annals of Operation Reserach*, 28:47–66, 1991.

[10] R. A. McCallum. Instance-based State Identification for Reinforcement Learning. In *Advances In Neural Information Processing Systems 7*, MIT Press, 1995.

[11] Edward J. Sondik. The optimal control of partially observable markov processes over the infinite horizon: Discounted costs. *Operations Reserach*, 26(2):282–304, 1978.

[12] R. S. Sutton. Learning to predict by the method of temporal differences. *Machine Learning*, 3:9–44, 1988.

[13] C. Watkins and P. Dayan. Q-learning. *Machine Learning*, 8:279–292, 1992.

[14] C. Wren, A. Azarbayejani, T. Darrell, and A. Pentland, Pfinder: Real-Time Tracking of the Human Body, Media Lab Per. Comp. TR-353, 1994
